# Greedy Algorithms for Structurally Constrained High Dimensional Problems

**Ambuj Tewari**
Department of Computer Science
University of Texas at Austin
ambuj@cs.utexas.edu

**Pradeep Ravikumar**
Department of Computer Science
University of Texas at Austin
pradeepr@cs.utexas.edu

**Inderjit S. Dhillon**
Department of Computer Science
University of Texas at Austin
inderjit@cs.utexas.edu

## Abstract

A hallmark of modern machine learning is its ability to deal with high dimensional problems by exploiting structural assumptions that limit the degrees of freedom in the underlying model. A deep understanding of the capabilities and limits of high dimensional learning methods under specific assumptions such as sparsity, group sparsity, and low rank has been attained. Efforts [1, 2] are now underway to distill this valuable experience by proposing general unified frameworks that can achieve the twin goals of summarizing previous analyses and enabling their application to notions of structure hitherto unexplored. Inspired by these developments, we propose and analyze a general computational scheme based on a greedy strategy to solve convex optimization problems that arise when dealing with structurally constrained high-dimensional problems. Our framework not only unifies existing greedy algorithms by recovering them as special cases but also yields novel ones. Finally, we extend our results to infinite dimensional settings by using interesting connections between smoothness of norms and behavior of martingales in Banach spaces.

## 1   Introduction

Increasingly in modern settings, in domains across science and engineering, one is faced with the challenge of working with high-dimensional models where the number of parameters is large, particularly when compared to the number of observations. In such high-dimensional regimes, a growing body of literature in machine learning and statistics has shown that it is typically impossible to obtain consistent estimators unless some low-dimensional "structure" is imposed on the high dimensional object that is being estimated from the data. For instance, the signal could be sparse in some basis, could lie on some manifold, have some graphical model structure, or be matrix-structured with a low rank. Indeed, given the variety of high dimensional problems that researchers face, it is natural that many novel notions of such low-dimensional structure will continue to appear in the future.

There are a variety of issues that researchers have grappled with in this area but two themes stand out. First, there is the *statistical* problem of identifying the minimum amount of data needed to accurately estimate high-dimensional objects that are structurally constrained. Second is the *computational* issue of designing *efficient* algorithms that, in the ideal case, can recover high dimensional objects from a limited amount of data. Both of these themes have spurred a huge amount of work over the past decade. For each of the specific structures, a large body of work has studied regularized and

constrained $M$-estimators, where some loss function such as the negative log-likelihood of the data which measures goodness of fit to the data, is regularized by a function appropriate to the assumed structure, or constrained to lie within an appropriately chosen set. In recent years, researchers [1, 2] studying the *statistical properties* of such estimators have started discovering commonalities among proofs and analyses and have proposed unified frameworks that take advantage of such commonalities. Specifically, using a single theorem, they are able to rederive a wide range of known results on high-dimensional consistency and error bounds for the various regularized and constrained estimators. The potential benefits are obvious: distillation of existing ideas and knowledge and the enabling of novel applications that are unexplored to date.

In this paper, we consider the *computational* facet of such high-dimensional estimation, and propose a general computational scheme that can be used for recovering objects with low-dimensional structure in the high dimensional setting. A key feature of our general method is that, at each step, it greedily chooses to add a single "simple element" or "atom" to the current representation. The idea, of course, is not new. Indeed we show that our general framework yields several existing greedy algorithms if we specialize it appropriately. It also yields novel algorithms that, to the best of our knowledge, have not appeared in the literature so far.

Greedy algorithms for optimizing smooth convex functions over the $\ell_1$-ball [3, 4, 5], the probability simplex [6] and the trace norm ball [7] have appeared in the recent literature. Other recent references on greedy learning algorithm for high-dimensional problems include [8, 9]. Greedy algorithms have also been studied in approximation theory [10, 11] to approximate a given function, viewed as an element of a Banach space of functions, using convex combinations of "simple" functions. There is also the well-known viewpoint of seeing *boosting algorithms* as greedy minimization algorithms in function space (see, for example, [12, Section 3], and the references therein). Often, the proofs and results in these various settings resemble each other to a great extent. There is thus clearly a need for unification of ideas and proofs.

In this paper, we focus on the underlying similarities between the greedy algorithms mentioned above. All these algorithms can be seen as specializations of a general computational scheme, with specific choices of the loss function, regularization or constraint set, and assumptions on the low-dimensional structure. Is there a commonality in their analyses of convergence rates, and are there key properties that inform such analyses? Here, we identify two such key properties. The first is a restricted smoothness property (RSP) parameter (see also [13], for a similar quantity), which relates the smoothness of the function when *restricted* to sets with low-dimensional structure, and which depends on the ambient space norm, as well as a potentially distinct norm in which smoothness is established. The other, established in [1, 2], measures the size of the low-dimensional structured object with respect to an "atomic" norm. Using these two quantities, we are able to provide a general theorem that yields convergence rates for general greedy methods. We recover a wide range of existing results, as well as some potentially novel ones, such as for block-sparse matrices, low-rank tensors, and permutation matrices. In certain cases, most notably for low rank tensors, the scheme appears to lead to a greedy step that is intractable, which leads to intriguing questions about tractable approximations that we hope will be adequately addressed in the future. We then show how to extend these results to a general infinite-dimensional setting, by extending our definition of the restricted smoothness property (RSP) parameter, which allows us to obtain rates for $L_p$ spaces as well Banach spaces with Martingale type $p$. For the latter, the RSP parameter hinges on the rate at which martingale difference sequences concentrate in that space, which provides yet another connection to the folk-lore statement that the "curse of dimensionality" in high dimensional settings is sometimes accompanied with the "blessings of concentration of measure".

## 2 Preliminaries

### 2.1 Atoms, Norms, and Structure

In Negahban et al.'s work[1], any specific structure such as sparsity is related to a low-dimensional subspace of structured vectors. In Chandrasekaran et al.'s work [2], this notion of structure is distilled further by the use of "atoms." Specifically, given a set $\mathcal{A}$ of very "simple" objects, called atoms, we can say that a vector $\mathbf{x}$ is simple (with some low-dimensional structure) if it can be written as a linear combination of few atoms: $\mathbf{x} = \sum_{i=1}^{k} c_i \mathbf{a}_i$, where $k$ is small relative to the ambient

dimensionality. They then use these atoms to generalize the idea behind the use of $\ell_1$-norm for sparsity, trace or nuclear norm for low rank, etc.

Let $\mathcal{A}$ be a collection of *atoms*. We start by assuming [2] that these atoms lie in a finite-dimensional space, and that in particular $\mathcal{A}$ is a compact subset of some Euclidean space $\mathbb{R}^p$. Later, in Section 6, we will extend our treatment to include the case where the atoms belong to an infinite-dimensional space. Let $\mathcal{C}_\mathcal{A}$ denote the *convex hull* of $\mathcal{A}$ and define the *gauge*:

$$\|\mathbf{x}\|_\mathcal{A} := \inf\{t \geq 0 \ : \ \mathbf{x} \in t\mathcal{C}_\mathcal{A})\} \ . \tag{1}$$

Note that the gauge $\| \cdot \|_\mathcal{A}$ is not a norm in general, unless for instance $\mathcal{A}$ satisfies a technical condition, namely that it be centrally symmetric: $\mathbf{x} \in \mathcal{A}$ iff $-\mathbf{x} \in \mathcal{A}$. Also, define the support function, $\|\mathbf{x}\|_\mathcal{A}^* := \sup\{\langle \mathbf{x}, \mathbf{a} \rangle \ : \ \mathbf{a} \in \mathcal{A}\}$. If $\| \cdot \|_\mathcal{A}$ happens to be a norm, then this is just the dual norm of $\| \cdot \|_\mathcal{A}$.

## 2.2 Examples

**Example 1. (Sparse vectors)** A huge amount of recent literature deals with the notion of sparsity of high-dimensional vectors. Here, the set $\mathcal{A} \subset \mathbb{R}^p$ of atoms is finite and consists of the $2p$ vectors $\pm\mathbf{e}_i$. This is a centrally symmetric set and hence $\| \cdot \|_\mathcal{A}$ becomes a norm, viz. the $\ell_1$-norm.

**Example 2. (Sparse non-negative vectors)** Using a slight variation on the previous example, the atoms can be the $p$ non-negative basis vectors $\mathbf{e}_i$. The convex hull $\mathcal{C}_\mathcal{A}$ is the $(p-1)$-dimensional probability simplex. This is not centrally symmetric and hence $\| \cdot \|_\mathcal{A}$ is not a norm.

**Example 3. (Group sparse matrices)** Here the structure we have in mind for a $p \times k$ matrix is that it only has a few non-zero *rows*. This generalizes Example 1 which can be thought of as the case when $k = 1$. There are an infinite number of atoms: all matrices with a single non-zero row where that row has $\ell_q$-norm 1 for some $q > 1$. The convex hull $\mathcal{C}_\mathcal{A}$ becomes the unit ball of the $\| \cdot \|_{q,1}$ group norm on $\mathbb{R}^{p \times k}$ that is defined to be the sum of the $\ell_q$-norms of the rows of its matrix argument.

**Example 4. (Low rank matrices)** This is another example that has attracted a huge amount of attention in the recent literature. The set [1] $\mathcal{A} \in \mathbb{R}^{p \times p}$ of atoms here is infinite and consists of rank-one matrices with Frobenius norm 1. This is centrally symmetric and $\| \cdot \|_\mathcal{A}$ becomes the trace norm (also called the *nuclear* or *Schatten*-1 norm, it is equal to the sum of the singular values of a matrix).

**Example 5. (Low rank tensors)** This is a generalization of the previous example to higher order tensors. Considering order three tensors, the set $\mathcal{A}$ of atoms can be taken to be all rank-one tensors of the form $\mathbf{u}_1 \otimes \mathbf{u}_2 \otimes \mathbf{u}_3 \in \mathbb{R}^{p \times p \times p}$ for $\mathbf{u}_i \in \mathbb{R}^p, \|\mathbf{u}_i\|_2 = 1$. Their convex hull is the unit ball of $\| \cdot \|_\mathcal{A}$ which can thought of as the tensor nuclear norm. Unfortunately, the tensor nuclear norm is intractable to compute and hence there is a need to consider relaxations to retain tractability.

**Example 6. (Permutation matrices)** Here, we consider permutation matrices[2] of size $p \times p$ as the set $\mathcal{A}$ of atoms. Even though there are $p!$ of them, their convex hull has a succinct description thanks to the *Birkhoff-von Neumann theorem*: the convex hull of permutation matrices is the set of doubly stochastic matrices. As we shall see later, this fact will be crucial for the greedy algorithm to be tractable in this case.

# 3 Problem Setup

We consider the general optimization problem

$$\min_{\mathbf{x} \ : \ \|\mathbf{x}\|_\mathcal{A} \leq \kappa} f(\mathbf{x}), \tag{2}$$

where $f$ is a *convex* and *smooth* function, and $\{\mathbf{x} \ : \ \|\mathbf{x}\|_\mathcal{A} \leq \kappa\}$ is the atomic norm constraint set that encourages some specific structure. This is a *convex optimization* problem that is a constrained version of the usual *regularized* problem, $\min_\mathbf{x} f(\mathbf{x}) + \mu\|\mathbf{x}\|_\mathcal{A}$. A line of recent work (see, for example, [2], and the references therein) has focused on different cases, with different atomic norms,

but largely on the linear case, where $f(\mathbf{x}) = \frac{1}{2}\|\mathbf{y} - \Phi\mathbf{x}\|_2^2$, for a given $\mathbf{y} \in \mathbb{R}^n$ and a linear map $\Phi : \mathbb{R}^p \to \mathbb{R}^n$. $\Phi$ is typically a linear measurement operator that generates a noisy measurement $\mathbf{y} \in \mathbb{R}^n$ from an underlying "simple" signal $\mathbf{x}_{\mathrm{tr}}$ and $\|\cdot\|_2$ is the standard Euclidean norm in $\mathbb{R}^n$. For the linear case, *projected gradient* type methods have been suggested [2]. In this paper, we consider the general problem in (2), with a general loss function $f(\mathbf{x})$, and a general constraint set induced by a structure-inducing atomic "norm" $\|\cdot\|_{\mathcal{A}}$.

### 3.1 Smoothness

We now discuss our assumptions on the loss function $f$ in (2). We start by defining a restricted smoothness property that we require for our analysis. Consider a convex function $f : \mathbb{R}^p \to \mathbb{R}$ that is differentiable on some convex subset $S$ of $\mathbb{R}^p$. Given a norm $\|\cdot\|$ on $\mathbb{R}^p$, we would like to measure how "smooth" the function $f$ is on $S$ with respect to $\|\cdot\|$. Towards this end, we define the following:

**Definition 1.** *Given a set $S$, and norm $\|\cdot\|$, we define the Restricted Smoothness Property (RSP) constant of a function $f : \mathbb{R}^p \to \mathbb{R}$ as*

$$L_{\|\cdot\|}\left(f; S\right) := \sup_{\mathbf{x},\mathbf{y} \in S, \alpha \in (0,1]} \frac{f((1-\alpha)\mathbf{x} + \alpha\mathbf{y}) - f(\mathbf{x}) - \langle \nabla f(\mathbf{x}), \alpha(\mathbf{y} - \mathbf{x})\rangle}{\alpha^2 \|\mathbf{y} - \mathbf{x}\|^2} . \tag{3}$$

Since $f$ is convex, it is clear that $L_{\|\cdot\|}\left(f; S\right) \geq 0$. The larger it is, the larger the function $f$ "curves up" on the set $S$.

**Remark 1.** *(Connection to Lipschitz continuity of the gradient)* Recall that a function $f : \mathbb{R}^p \to \mathbb{R}$ is said to have $L$-Lipschitz continuous gradients w.r.t. $\|\cdot\|$ if for all $\mathbf{x}, \mathbf{y} \in \mathbb{R}^p$, we have $\|\nabla f(\mathbf{x}) - \nabla f(\mathbf{y})\|^* \leq L \cdot \|\mathbf{x} - \mathbf{y}\|$ where $\|\cdot\|^*$ is the norm dual to $\|\cdot\|$. Using the mean value theorem it is easy to see that if $f$ has $L$-Lipschitz continuous gradient w.r.t. $\|\cdot\|$ then $L_{\|\cdot\|}\left(f; S\right) \leq L$. However, $L_{\|\cdot\|}\left(f; S\right)$ can be much smaller since it only looks at the behavior of $f$ on $S$ and cares less about the global smoothness of $f$.

**Remark 2.** *(Connection to boundedness of the Hessian)* If the function $f$ is twice differentiable on $S$, using second order Taylor expansion, $L_{\|\cdot\|}\left(f; S\right)$ can be bounded as

$$L_{\|\cdot\|}\left(f; S\right) \leq \sup_{\mathbf{x},\mathbf{y},\mathbf{z} \in S} \frac{\langle \nabla^2 f(\mathbf{z})(\mathbf{y} - \mathbf{x}), \mathbf{y} - \mathbf{x}\rangle}{\|\mathbf{y} - \mathbf{x}\|^2} . \tag{4}$$

Again, suppose we have *global* control on $\nabla^2 f(\mathbf{x})$ in the form $\forall \mathbf{z} \in \mathbb{R}^p$, $\|\|\nabla^2 f(\mathbf{z})\|\| \leq H$ where $\|\|\cdot\|\|$ is the $\|\cdot\| \to \|\cdot\|_\star$ operator norm of the matrix $M$ defined as $\|\|M\|\| := \sup_{\|\mathbf{x}\| \leq 1} \|M\mathbf{x}\|_\star$. Then, we immediately have $L_{\|\cdot\|}\left(f; S\right) \leq H$ but this inequality might be loose in general.

In the statement of our results, we will derive convergence rates that would depend on this Restricted Smoothness Property (RSP) constant of the loss function $f$ in (2).

## 4 Greedy Algorithm and Analysis

In this section, we consider a general greedy scheme to solve the general optimization problem in (2) where $f$ is a *convex, smooth* function. The idea is to add one atom to our representation at a time in a way that the stucture of the set of atoms can be exploited to perform the greedy step efficiently. Our greedy method is applicable to any constrained problem where the objective is sufficiently *smooth*.

---

**Algorithm 1** A general greedy algorithm to minimize a convex function $f$ over the $\kappa$-scaled atomic-norm "ball"

---
1: $\mathbf{x}_0 \leftarrow \kappa \mathbf{a}_0$ for an arbitrary atom $\mathbf{a}_0 \in \mathcal{A}$
2: **for** $t = 0, 1, 2, 3, \ldots$ **do**
3:     $\mathbf{a}_t \leftarrow \mathrm{argmin}_{\mathbf{a} \in \mathcal{A}} \langle \nabla f(\mathbf{x}_t), \mathbf{a}\rangle$
4:     $\alpha_t \leftarrow \mathrm{argmin}_{\alpha \in [0,1]} f(\mathbf{x}_t + \alpha(\kappa \mathbf{a}_t - \mathbf{x}_t))$
5:     $\mathbf{x}_{t+1} \leftarrow \mathbf{x}_t + \alpha_t(\kappa \mathbf{a}_t - \mathbf{x}_t)$
6: **end for**

---

**Theorem 1.** *Assume that $f$ is convex and differentiable and let $\|\cdot\|$ be any norm. Then, for any $T \geq 1$, the iterates generated by Algorithm 1 lie in $\kappa \mathcal{C}_\mathcal{A}$ and satisfy,*

$$f(\mathbf{x}_T) - f(\mathbf{x}^\star) \leq \frac{8\kappa^2 \cdot L_{\|\cdot\|}(f; \kappa\mathcal{C}_\mathcal{A}) \cdot \|\mathcal{A}\|^2}{T}, \tag{5}$$

*for any solution $\mathbf{x}^\star$ of (2). Here $L_{\|\cdot\|}(f; \kappa\mathcal{C}_\mathcal{A})$ is the smoothness constant as defined in (3) and $\|\mathcal{A}\| := \sup_{\mathbf{a} \in \mathcal{A}} \|\mathbf{a}\|$.*

*Proof.* Let us use the abbreviations $L$ and $R$ for $L_{\|\cdot\|}(f; S)$ and $\|\mathcal{A}\|$ respectively. The fact that the iterates lie in $\kappa\mathcal{C}_\mathcal{A}$ follows immediately from the definition of the algorithm and a simple induction. Now assuming $\mathbf{x}_t \in \kappa\mathcal{C}_\mathcal{A}$, we have, by definition of $L$, for any $\alpha \in [0,1]$,

$$
\begin{aligned}
f(\mathbf{x}_t + \alpha(\kappa\mathbf{a}_t - \mathbf{x}_t)) &\leq f(\mathbf{x}_t) + \alpha\langle\nabla f(\mathbf{x}_t), \kappa\mathbf{a}_t - \mathbf{x}_t\rangle + \tfrac{1}{2}\alpha^2 L\|\kappa\mathbf{a}_t - \mathbf{x}_t\|^2 \\
&\leq f(\mathbf{x}_t) + \alpha\langle\nabla f(\mathbf{x}_t), \kappa\mathbf{a}_t - \mathbf{x}_t\rangle + \tfrac{1}{2}\alpha^2 L\left(2\|\kappa\mathbf{a}_t\|^2 + 2\|\mathbf{x}_t\|^2\right) \\
&\leq f(\mathbf{x}_t) - \alpha(-\langle\nabla f(\mathbf{x}_t), \kappa\mathbf{a}_t\rangle + \langle\nabla f(\mathbf{x}_t), \mathbf{x}_t\rangle) + 2\alpha^2 L\kappa^2 R^2 \ . \tag{6}
\end{aligned}
$$

The last inequality holds because $\|\kappa\mathbf{a}_t\|, \|\mathbf{x}_t\| \leq \kappa R$. Now, for any minimizer $\mathbf{x}^\star$ of $f$, we have, by convexity of $f$,

$$
\begin{aligned}
\delta_t := f(\mathbf{x}_t) - f(\mathbf{x}^\star) &\leq \langle\nabla f(\mathbf{x}_t), \mathbf{x}_t - \mathbf{x}^\star\rangle = \langle\nabla f(\mathbf{x}_t), \mathbf{x}_t\rangle - \langle\nabla f(\mathbf{x}_t), \mathbf{x}^\star\rangle \\
&\leq \langle\nabla f(\mathbf{x}_t), \mathbf{x}_t\rangle - \langle\nabla f(\mathbf{x}_t), \kappa\mathbf{a}_t\rangle \ . \tag{7}
\end{aligned}
$$

The last inequality holds because, $\mathbf{a}_t$ is the minimizer of the linear function $\langle\nabla f(\mathbf{x}_t), \cdot\rangle$ over $\mathcal{A}$ (and hence also over $\mathcal{C}_\mathcal{A}$) and $\mathbf{x}^\star/\kappa \in \mathcal{C}_\mathcal{A}$. Thus, $\langle\nabla f(\mathbf{x}_t), \mathbf{a}_t\rangle \leq \langle\nabla f(\mathbf{x}_t), \mathbf{x}^\star/\kappa\rangle$. Plugging (7) into (6), we have, for any $\alpha \geq 0$, $f(\mathbf{x}_t + \alpha(\kappa\mathbf{a}_t - \mathbf{x}_t)) \leq f(\mathbf{x}_t) - \alpha\delta_t + 2\alpha^2 L\kappa^2 R^2$. Since $\mathbf{x}_{t+1}$ is chosen by minimizing the LHS over $\alpha \in [0,1]$, we have $f(\mathbf{x}_{t+1}) \leq f(\mathbf{x}_t) + \min_{\alpha \in [0,1]}(-\alpha\delta_t + 2\alpha^2 L\kappa^2 R^2)$. Thus, we have, for all $t \geq 0$, $\delta_{t+1} \leq \delta_t + \min_{\alpha \in [0,1]}(-\alpha\delta_t + 2\alpha^2 L\kappa^2 R^2)$. For $t = 0$, choose $\alpha = 1$ on the RHS to get $\delta_1 \leq 2L\kappa^2 R^2$. Since $\delta_t$'s are decreasing, this shows $\delta_t \leq 2L\kappa^2 R^2$ for all $t \geq 1$. Hence, for $t \geq 1$, we can choose $\alpha = \delta_t/4L\kappa^2 R^2 \in [0, \frac{1}{2}]$ on the RHS to get $\forall t \geq 1$, $\delta_{t+1} \leq \delta_t - \frac{\delta_t^2}{8L\kappa^2 R^2}$. Solving this recursion easily gives, for all $t \geq 1$, $f(\mathbf{x}_{t+1}) - f(\mathbf{x}^\star) \leq \frac{8\kappa^2 \cdot L \cdot R^2}{t}$. $\square$

**Remark 3.** We emphasize that the norm $\|\cdot\|$ appears *only in the analysis* and *not* in the algorithm. Since the bound of Theorem 1 is *simultaneously* true for all norms $\|\cdot\|$, the best bound is achieved by choosing a norm that minimizes the product of $\|\mathcal{A}\|^2$ and $L_{\|\cdot\|}(f; \kappa\mathcal{C}_\mathcal{A})$.

**Remark 4.** We make the simple but useful observation that the iterate $\mathbf{x}_t$ can be written as a convex combination of at most $t + 1$ atoms, namely $\mathbf{a}_0, \mathbf{a}_1, \ldots, \mathbf{a}_t$.

**Remark 5.** Given $\kappa$, Algorithm 1 is completely parameter free. This is a nice feature from a practical perspective as it frees the practitioner from the task of tuning parameters.

## 5 Special Cases

Let us revisit the examples from Section 2.2 to see what concrete algorithms and accuracy bounds we get by specializing Algorithm 1 and its bound (Theorem 1) to them.

**Sparse vectors** The greedy step reduces to

$$\mathbf{a}_t \leftarrow \operatorname*{argmin}_{\mathbf{a} \in \pm\{\mathbf{e}_1, \ldots, \mathbf{e}_p\}} \langle\nabla f(\mathbf{x}_t), \mathbf{a}\rangle \ .$$

Clearly, assuming that the gradient is already available, this can be done in $O(p)$ time by finding $j \in \{1, \ldots, p\}$ such that $j = \operatorname{argmax}_{j'} |[\nabla f(\mathbf{x}_t)]_{j'}|$ and setting $\mathbf{a}_t = -\operatorname{sign}([\nabla f(\mathbf{x}_t)]_j)\mathbf{e}_j$. This actually gives a well-known algorithm whose roots go back to the 1950s [3]. More recently, a variant appeared as the *Forward Greedy Selection* algorithm in [5] (see also [4]). In fact, the original Frank-Wolfe algorithm can be applied whenever the set $\mathcal{C}_\mathcal{A}$ is polyhedral. If we choose the norm $\|\cdot\|$ to be $\ell_q$ then $\|\mathcal{A}\|$ is 1 irrespective of $q \in [1, \infty]$ and the smoothness constant $L_{\|\cdot\|_q}(f; \kappa\mathcal{C}_\mathcal{A})$ is an increasing function of $q$. Hence to minimize the bound, we should choose $p = 1$ and measure smoothness of $f$ over the $\kappa$-scaled $\ell_1$-ball using the $\ell_1$-norm. When $f(\mathbf{x}) = \frac{1}{2}\|\mathbf{y} - \Phi\mathbf{x}\|_2^2$, we can use the connection to Hessian bounds (Remark 2) and immediately get the upper bound $8\kappa^2 \cdot \|\Phi^\top\Phi\|_{1\to\infty}/T$ where the norm $\|M\|_{1\to\infty} := \sup_{\|\mathbf{x}\|_1 \leq 1} \|M\mathbf{x}\|_\infty$ is simply $\max_{i,j} |M_{i,j}|$.

**Sparse non-negative vectors** The greedy step becomes

$$\mathbf{a}_t \leftarrow \operatorname*{argmin}_{\mathbf{a} \in \{\mathbf{e}_1, \dots, \mathbf{e}_p\}} \langle \nabla f(\mathbf{x}_t), \mathbf{a} \rangle \ .$$

As in the previous example, this can be done in $O(p)$ time given the gradient entries by computing $j = \operatorname{argmin}_{j' \in \{1, \dots, p\}} [\nabla f(\mathbf{x}_t)]_{j'}$ and setting $\mathbf{a}_t = \mathbf{e}_j$. This particular algorithm to optimize a smooth function over the (scaled) probability simplex appears in [6]. Following the same reasoning as above, we get the best (among all $\ell_q$-norms) bound if we choose $\| \cdot \|$ to be $\| \cdot \|_1$ and then our smoothness constant becomes similar to Clarkson's "nonlinearity measure" that he denotes by $C_f$.

**Group sparse matrices** This is an interesting case since there are an infinite number of atoms. But still the greedy step

$$\mathbf{a}_t \leftarrow \operatorname*{argmin}_{\mathbf{a} \,:\, \text{nnzrows}(\mathbf{a})=1, \|\mathbf{a}\|_{q,1}=1} \langle \nabla f(\mathbf{x}_t), \mathbf{a} \rangle$$

(where nnzrows counts the number of non-zero rows of a matrix) can be computed easily as follows. Let $q'$ be the dual exponent of $q$ that satisfies $1/q + 1/q' = 1$ and find the row $j$ of $\nabla f(\mathbf{x}_t)$ with maximal $\ell_{q'}$ norm. Then, set $\mathbf{a}_t$ to be the matrix all of whose rows are zero except row $j$. In row $j$, place the vector $\mathbf{u}^\top$ where $\mathbf{u} \in \mathbb{R}^{k \times 1}$ is such that[3] $\langle \mathbf{u}, [\nabla f(\mathbf{x}_t)]_{j,:}^\top \rangle = -\|[\nabla f(\mathbf{x}_t)]_{j,:}^\top\|_{q'}$ and $\|\mathbf{u}\|_q = 1$. Such a vector $\mathbf{u}$ can be found in closed form. For the case $f(\mathbf{x}_t) = \frac{1}{2}\|\mathbf{y} - \Phi\mathbf{x}\|_F^2$, choosing the norm $\| \cdot \|$ in Theorem 1 to be $\| \cdot \|_{q,1}$ (and this gives the optimal bound among all $\| \cdot \|_{q,r}$ norms for $r > 1$), we get the accuracy bound: $8\kappa^2 \cdot \|\Phi^\top \Phi\|_{q,1 \to q,\infty}/T$ where the $q, 1 \to q, \infty$ norm of the operator $\Phi^\top \Phi$ is defined as $\sup\{\|\Phi^\top \Phi M\|_{q,\infty} : M \in \mathbb{R}^{p \times k}, \|M\|_{q,1} \leq 1\}$. This algorithm and its analysis are novel to the best of our knowledge. However, we note that a related greedy algorithm (that does not directly optimize the objective (2)) called *Group-OMP* appears in [14, 15].

**Low rank matrices** As in the previous case, we have an infinite number of atoms: all rank-1 matrices with Frobenius norm 1. Yet, the greedy step

$$\mathbf{a}_t \leftarrow \operatorname*{argmin}_{\mathbf{a} \,:\, \text{rank}(\mathbf{a})=1, \|\mathbf{a}\|_F=1} \langle \nabla f(\mathbf{x}_t), \mathbf{a} \rangle$$

can be done in polynomial time by computing the SVD, $\nabla f(\mathbf{x}_t) = U\Sigma V^\top$ and setting $\mathbf{a} = -\mathbf{u}_1 \mathbf{v}_1^\top$ where $\mathbf{u}_1, \mathbf{v}_1$ are the left, right singular vectors corresponding to the largest singular value $\sigma_1$. Since we only need the singular vectors corresponding to the largest singular value, the computation of $\mathbf{a}_t$ can be done much faster than the time it takes to compute a full SVD. For the case $f(\mathbf{x}) = \frac{1}{2}\|\mathbf{y} - \Phi\mathbf{x}\|_F^2$, the bound of Theorem 1 is minimized, among all *Schatten-p* norms[4], by using $\| \cdot \| = \| \cdot \|_{S(1)}$, i.e. the trace or nuclear norm. Since the objective is twice differentiable, using Remark 2 we get the following upper bound on the accuracy: $8\kappa^2 \cdot \|\Phi^\top \Phi\|_{S(1) \to S(\infty)}/T$ which depends on the $S(1) \to S(\infty)$ operator norm of $\Phi^\top \Phi$ which is defined as $\sup\{\|\Phi^\top \Phi M\|_{S(\infty)} : M \in \mathbb{R}^{p \times p}, \|M\|_{S(1)} \leq 1\}$. This algorithm was recently independently discovered and analyzed in [7].

**Low rank tensors** Here, the greedy step

$$\mathbf{a}_t \leftarrow \operatorname*{argmin}_{\mathbf{a} \,:\, \mathbf{a}=\mathbf{u}_1 \otimes \mathbf{u}_2 \otimes \mathbf{u}_3, \|\mathbf{u}_i\|_2=1} \langle \nabla f(\mathbf{x}_t), \mathbf{a} \rangle$$

appears intractable. Indeed, the above problem is closely related to the problem of finding the *best rank-one approximation* to a given tensor which is known to be NP-hard [16] already for order-3 tensors. However, as described in [2], it is possible to construct a family of *outer approximations* $\mathcal{C}_\mathcal{A} \subseteq \dots \subseteq TH_{k+1} \subseteq TH_k$ such that, for any fixed $k$, $TH_k$ can be described by a *semidefinite program* of size polynomial in $k$. So, even though the exact greedy step above may not be tractable, we can use these "theta bodies" (whence the notation "$TH$") to approximate the greedy step. The iterates will no longer lie strictly in the tensor nuclear ball of the given radius. Understanding the implications of such approximations and their analysis are interesting questions to pursue but lie beyond the scope of the current paper.

**Permutation matrices** Here, fortunately, we again do not face intractability: the step

$$\mathbf{a}_T \leftarrow \underset{\mathbf{a}\,:\,\mathbf{a}\text{ is a permutation matrix}}{\operatorname{argmin}} \langle \nabla f(\mathbf{x}_t), \mathbf{a} \rangle$$

reduces to solving a *linear assignment problem* with costs $C(i,j) = [\nabla f(\mathbf{x}_t)]_{i,j}$. This can be efficiently done using, for example, the Hungarian algorithm. Another way to see that the above step does not involve combinatorial explosion is to appeal to the *Birkhoff-von Neumann* theorem that states that the convex hull of permutation matrices is the set of doubly stochastic matrices. As a result, the above reduces to minimizing a linear objective $\langle \nabla f(\mathbf{x}_t), M \rangle$ subject to polynomially many constraints: $M \geq 0$, $M\mathbf{1} = \mathbf{1}$ and $M^\top \mathbf{1} = \mathbf{1}$.

# 6 Extension to Infinite Dimensional Banach Spaces

In thi section, we consider an extension of the framework behind Algorithm 1 to the case when the set of atoms are in some infinite dimensional (real) Banach space $(V, \| \cdot \|)$. For example, the atoms could be some "simple" real valued functions on some interval $[a, b] \subseteq \mathbb{R}$. The two ingredients in our framework were the atomic norms, and the Restricted Smoothness Property (RSP) parameters. In [2], and in Section 2.1, the atoms were considered as belonging to a finite dimensional Euclidean space. Note however that the definition of the atomic norms in (1) did not make use of the topology of the ambient space, and hence is applicable even when the atoms belong to some Banach space $(V, \| \cdot \|)$. However, our definition of the RSP parameter in (3) relied critically on the Euclidean inner product, whence we will now extend this to the infinite dimensional case in the sequel.

Consider a convex continuous Fréchet differentiable function $f : V \to \mathbb{R}$, and let $\nabla f(\mathbf{x})$ denote the Fréchet derivative of $f$ at $\mathbf{x}$. Let $\langle \cdot, \cdot \rangle : V^* \times V \to \mathbb{R}$ denotes the bilinear function (which is *not an inner product* in general) $\langle \mathbf{X}, \mathbf{x} \rangle := \mathbf{X}(\mathbf{x})$ for $\mathbf{x} \in V$ and $\mathbf{X}$ in the dual space $V^*$ (consisting of bounded linear functions on $V$).

**Definition 2.** *Given a Banach space $(V, \| \cdot \|)$, and a set $S \subseteq V$, and some $r \in [1, 2]$, we define the Restricted Uniform Smoothness Property (RUSP) constant of a convex continuous Fréchet differentiable function $f : V \to \mathbb{R}$ as*

$$L_r(f; S) := \sup_{\mathbf{x}, \mathbf{y} \in S, \alpha \in [0,1]} \frac{f((1-\alpha)\mathbf{x} + \alpha\mathbf{y}) - f(\mathbf{x}) - \langle \nabla f(\mathbf{x}), \alpha(\mathbf{y} - \mathbf{x}) \rangle}{(1/r)\,\alpha^r \|\mathbf{y} - \mathbf{x}\|^r}. \tag{8}$$

This need not be bounded in general, but would be bounded for instance if the function $f$ were $r$-uniformly smooth (though this would be a far stronger condition). Suppose the set of atoms $\mathcal{A} \subseteq V$ is such that $\max_{\mathbf{a} \in \mathcal{A}} \langle \mathbf{X}, \mathbf{a} \rangle$ is defined for any $\mathbf{X} \in V^*$. Then, we can define a straightforward extension of Algorithm 1 given as Algorithm 2.

---

**Algorithm 2** A general greedy algorithm to minimize a continuous Fréchet differentiable convex function $f$ over the convex hull of a set of atoms $\mathcal{A}$ in a Banach space $(V, \| \cdot \|)$

---

1: $\mathbf{x}_0 \leftarrow \mathbf{a}_0$ for an arbitrary atom $\mathbf{a}_0 \in \mathcal{A}$
2: **for** $t = 0, 1, 2, 3, \ldots$ **do**
3:     $\mathbf{X}_t \in V^* \leftarrow \nabla f(\mathbf{x}_t)$, the Fréchet derivative of $f$ at $\mathbf{x}_t$
4:     $\mathbf{a}_t \leftarrow \operatorname{argmax}_{\mathbf{a} \in \mathcal{A}} \langle -\mathbf{X}_t, \mathbf{a} \rangle$
5:     $\alpha_t \leftarrow \operatorname{argmin}_{\alpha \in [0,1]} f(\mathbf{x}_t + \alpha(\mathbf{a}_t - \mathbf{x}_t))$
6:     $\mathbf{x}_{t+1} \leftarrow \mathbf{x}_t + \alpha_t(\mathbf{a}_t - \mathbf{x}_t)$
7: **end for**

---

The following result proves a general rate of convergence for Algorithm 2. Since the proof follows the proof of Theorem 1 very closely, we defer it to the appendix.

**Theorem 2.** *Suppose that $(V, \| \cdot \|)$ is a Banach space and let $f : V \to \mathbb{R}$ be a convex continuous Fréchet differentiable function. Let $\mathcal{A}$ be a set of atoms such that $\|\mathbf{a}\| \leq R$ for all $\mathbf{a} \in \mathcal{A}$, and let $S = \operatorname{conv}(\mathcal{A})$. Suppose the Restricted Uniform Smoothness Property (RUSP) constant $L_r(f; S)$ of $f$ is bounded for some $r \in [1, 2]$. Then,*

$$f(\mathbf{x}_t) - \inf_{\mathbf{x} \in S} f(\mathbf{x}) = O\left(\frac{L_r(f; S)\, R^r}{t^{r-1}}\right)$$

*where the hidden constant depends on $r$ only.*

## 6.1 Rates of Convex Approximation in $L_p$ spaces

For $p \in (1, \infty)$ the space $L_p([a, b])$ consists of all functions $g : [a, b] \rightarrow \mathbb{R}$ such that the (Lebesgue) integral $\int_a^b |g(x)|^p dx$ is finite. The space $L_p$ is a Banach space once we equip it with the norm $\|g\|_{L_p} := \left( \int_a^b |g(x)|^p dx \right)^{1/p}$. Let $\mathcal{A}$ be a set of atoms in $L_p$ with bounded norm and let $h \in L_p$ be a function that we wish to approximate using convex combinations of the atoms. Since, the function $g \mapsto \|g\|_{L_p}^{p'}$ is $p' = \min\{p, 2\}$ uniformly smooth for $p \in (1, \infty)$, we can use Algorithm 2 to generate a sequence of functions $g_1, g_2, \ldots$ such that $g_t$ is a convex combination of only $t$ atoms. Moreover, we will have the guarantee: $\|g_{t+1} - h\|_{L_p}^{p'} - \inf_{g \in \text{conv}(\mathcal{A})} \|g - h\|_{L_p}^{p'} = O\left( \frac{1}{t^{p'-1}} \right)$. Such rates of convex approximation in non-Hilbert spaces have been studied earlier (see, for example, [10, 11]). Note that, unlike [10], we do not assume that $h \in \text{conv}(\mathcal{A})$. If that is the case, the above rate simplifies to the rates given in [10]: $O(t^{-1+\frac{1}{p}})$ for $p \in (1, 2)$, and $O(t^{-\frac{1}{2}})$ for $p > 2$.

## 6.2 Rates of Convex Approximation in Spaces with Martingale Type $p$

Note the fact that, in the previous subsection, the only property of $L_p$ spaces that we used to get rates was the fact that the norm to some power was a uniformly smooth convex function. It turns out that the existence of uniformly smooth functions in a given Banach space is intimately connected to the behavior of martingale difference sequences in that space. To precisely state the connection, we need to define the notion of *martingale type* (also called *Haar type*) [17, p. 320]. A Banach space $(V, \| \cdot \|)$ is said to have martingale type $p$ (M-type $p$ in short) if there exists a constant $K_p$ such that, for all $T \geq 1$, and any $V$-valued martingale difference sequence $\mathbf{d}_1, \ldots, \mathbf{d}_T$, we have

$$\mathbb{E}\left[ \left\| \sum_{t=1}^{T} \mathbf{d}_t \right\| \right] \leq K_p \cdot \left( \sum_{t=1}^{p} \mathbb{E}\left[ \|\mathbf{d}_t\|^p \right] \right)^{1/p}.$$

Note that, by triangle inequality for norms, any Banach space always has M-type 1 while a Hilbert space (i.e. the norm $\| \cdot \|$ comes from an inner product) has M-type 2. Hilbert space essentially have the best M-type in the sense that no Banach space has M-type $p$ for $p > 2$. The connection of M-type to uniform smoothness is made precise by the following remarkable theorem (see also [18]).

**Theorem (Pisier, [19].** *A Banach space has M-type $p$ iff there is an equivalent norm[5] $\|\cdot\|_\#$ such that the function $\|\cdot\|_\#^p$ is $p$-uniformly smooth.*

Consider the setting of the previous subsection where we have some $h \in \text{conv}(\mathcal{A})$ for some set $\mathcal{A}$ of atoms in an arbitrary Banach space $(V, \| \cdot \|)$. Using Pisier's theorem, we get the following corollary.

**Corollary 3.** *Suppose $\mathcal{A}$ is a set of atoms in a Banach space $(V, \| \cdot \|)$ that has M-type $p$ and let $h \in \text{conv}(\mathcal{A})$. Suppose Algorithm 2 generates iterates $g_1, g_2, \ldots$ when run on the function $g \mapsto \|g\|_\#^p$ whose existence is guaranteed by Pisier's theorem. Then, we have, $\|g_{t+1} - h\| = O\left( t^{-1+\frac{1}{p}} \right)$.*

## 7 Future Work

First, we envisage the algorithm being used to compute the *entire regularization path* corresponding to all values of the constraint parameter $\kappa$. Using a *warm start* strategy, where the algorithm for higher values of $\kappa$ is initialized with the solution for lower values, can be very helpful here. Exploring this to get a general practical algorithm to compute the entire path would be very nice. Third, linear convergence guarantees for *projected gradient* type methods have been obtained by [13] where they make the additional assumption of (generalized) restricted *strong convexity*. It should be possible to derive similar faster rates for our greedy algorithm.

### Acknowledgments

We gratefully acknowledge the support of NSF under grant IIS-1018426. ISD acknowledges support from the Moncrief Grand Challenge Award.

## Footnotes

[1] For simplicity we consider square matrices. It is definitely also possible to consider rectangular matrices in $\mathbb{R}^{p_1 \times p_2}$ for $p_1 \neq p_2$

[2] A *permutation matrix* is one consisting only of 0's & 1's such that there is exactly a single 1 in each row & column. A non-negative matrix with every row & column sum equal to 1 is called a *doubly stochastic matrix*.

[3]We use MATLAB notation $M_{j,:}$ to denote row $j$ of a matrix $M$.

[4]The Schatten-$q$ norm of a matrix is the $\ell_q$ norm of its singular values.

[5] That is, $c_L \| \cdot \| \leq \|\cdot\|_\# \leq c_U \| \cdot \|$ for some $c_L, c_U > 0$.

# References

[1] S. Negahban, P. Ravikumar, M. Wainwright, and B. Yu. A unified framework for high-dimensional analysis of M-estimators with decomposable regularizers. In *Advances in Neural Information Processing Systems 22*, pages 1348–1356, 2009.

[2] V. Chandrasekaran, B. Recht, P. A. Parrilo, and A. S. Willsky. The convex geometry of linear inverse problems. In *Proceedings of the 48th Annual Allerton Conference on Communication, Control and Computing*, pages 699–703, 2010.

[3] M. Frank and P. Wolfe. An algorithm for quadratic programming. *Naval Research Logistics Quarterly*, 3(1-2):95–110, 1956.

[4] T. Zhang. Sequential greedy approximation for certain convex optimization problems. *IEEE Transactions on Information Theory*, 49(3):682–691, 2003.

[5] S. Shalev-Shwartz, N. Srebro, and T. Zhang. Trading accuracy for sparsity in optimization problems with sparsity constraints. *SIAM Journal on Optimization*, 20(6):2807–2832, 2010.

[6] K. Clarkson. Coresets, sparse greedy approximation, and the Frank-Wolfe algorithm. In *Proceedings of the nineteenth annual ACM-SIAM symposium on discrete algorithms*, pages 922–931. Society for Industrial and Applied Mathematics, 2008.

[7] S. Shalev-Shwartz, A. Gonen, and O. Shamir. Large-scale convex minimization with a low-rank constraint. In *Proceedings of the 28th International Conference on Machine Learning*, pages 329–336, 2011.

[8] H. Liu and X. Chen. Nonparametric greedy algorithms for the sparse learning problem. In *Advances in Neural Information Processing Systems 22*, pages 1141–1149, 2009.

[9] A. Das and D. Kempe. Submodular meets spectral: Greedy algorithms for subset selection, sparse approximation and dictionary selection. In *Proceedings of the 28th International Conference on Machine Learning*, pages 1057–1064, 2011.

[10] M. J. Donahue, C. Darken, L. Gurvits, and E. Sontag. Rates of convex approximation in non-Hilbert spaces. *Constructive Approximation*, 13(2):187–220, 1997.

[11] V. N. Temlyakov. Greedy approximation. *Acta Numerica*, 17:235–409, 2008.

[12] R. E. Schapire. The boosting approach to machine learning: an overview. In D. D. Denison, M. H. Hansen, C. C. Holmes, B. Mallick, and B. Yu, editors, *Nonlinear estimation and classification*, volume 171 of *Lecture Notes in Statistics*, pages 149–172. Springer, 2003.

[13] A. Agarwal, S. Negahban, and M. Wainwright. Fast global convergence rates of gradient methods for high-dimensional statistical recovery. In *Advances in Neural Information Processing Systems 23*, pages 37–45, 2010.

[14] A. C. Lozano, G. Świrszcz, and N. Abe. Grouped orthogonal matching pursuit for variable selection and prediction. In *Advances in Neural Information Processing Systems 22*, pages 1150–1158, 2009.

[15] A. C. Lozano, G. Świrszcz, and N. Abe. Grouped orthogonal matching pursuit for logistic regression. In *Proceedings of the Fourteenth International Conference on Artificial Intelligence and Statistics*, volume 15 of *JMLR Workshop and Conference Proceedings*, 2011.

[16] C. Hillar and L.-H. Lim. Most tensor problems are NP hard, 2010. available at http://arxiv.org/abs/0911.1393v2.

[17] A. Pietsch. *History of Banach spaces and linear operators*. Birkhäuser, 2007.

[18] J. Borwein, A. J. Guirao, P. Hájek, and J. Vanderwerff. Uniformly convex functions in Banach spaces. *Proceedings of the American Mathematical Society*, 137(3):1081–1091, 2009.

[19] G. Pisier. Martingales with values in uniformly convex spaces. *Israel Journal of Mathematics*, 20(3-4):326–350, 1975.

